# Quasi-Newton Methods
# for Markov Chain Monte Carlo

**Yichuan Zhang and Charles Sutton**
School of Informatics
University of Edinburgh
Y.Zhang-60@sms.ed.ac.uk, csutton@inf.ed.ac.uk

## Abstract

The performance of Markov chain Monte Carlo methods is often sensitive to the scaling and correlations between the random variables of interest. An important source of information about the local correlation and scale is given by the Hessian matrix of the target distribution, but this is often either computationally expensive or infeasible. In this paper we propose MCMC samplers that make use of *quasi-Newton* approximations, which approximate the Hessian of the target distribution from previous samples and gradients generated by the sampler. A key issue is that MCMC samplers that depend on the history of previous states are in general not valid. We address this problem by using *limited memory* quasi-Newton methods, which depend only on a fixed window of previous samples. On several real world datasets, we show that the quasi-Newton sampler is more effective than standard Hamiltonian Monte Carlo at a fraction of the cost of MCMC methods that require higher-order derivatives.

## 1 Introduction

The design of effective approximate inference methods for continuous variables often requires considering the curvature of the target distribution. This is especially true of Markov chain Monte Carlo (MCMC) methods. For example, it is well known that the Gibbs sampler mixes extremely poorly on distributions that are strongly correlated. In a similar way, the performance of a random walk Metropolis-Hastings algorithm is sensitive to the variance of the proposal distribution. Many samplers can be improved by incorporating second-order information about the target distribution. For example, several authors have used a Metropolis-Hastings algorithm in which the Hessian is used to form a covariance for a Gaussian proposal [3, 11]. Recently, Girolami and Calderhead [5] have proposed a Hamiltonian Monte Carlo method that can require computing higher-order derivatives of the target distribution.

Unfortunately, second derivatives can be inconvenient or infeasible to obtain and the quadratic cost of manipulating a $d \times d$ Hessian matrix can also be prohibitive. An appealing idea is to approximate the Hessian matrix using the sequence of first order information of previous samples, in a manner similar to *quasi-Newton* methods from the optimization literature. However, samplers that depend on the history of previous samples must be carefully designed in order to guarantee the chain converges to the target distribution.

In this paper, we present quasi-Newton methods for MCMC that are based on approximations to the Hessian from first-order information. In particular, we present a Hamiltonian Monte Carlo algorithm in which the variance of the momentum variables is based on a BFGS approximation. The key point is that we use a *limited memory* approximation, in which only a small window of previous samples are used to the approximate the Hessian. This makes it straightforward to show that our samplers are valid, because the samples are distributed as a order-$k$ Markov chain. Second, by taking advantage

of the special structure in the Hessian approximation, the samplers require only linear time and linear space in the dimensionality of the problem. Although this is a very natural approach, we are unaware of previous MCMC methods that use quasi-Newton approximations. In general we know of very few MCMC methods that make use of the rich set of approximations from the numerical optimization literature (some exceptions include [7, 11]). On several logistic regression data sets, we show that the quasi-Newton samplers produce samples of higher quality than standard HMC, but with significantly less computation time than methods that require higher-order derivatives.

## 2 Background

In this section we provide background on Hamiltonian Monte Carlo. An excellent recent tutorial is given by Neal [9]. Let $\mathbf{x}$ be a random variable on state space $\mathcal{X} = \mathbb{R}^d$ with a target probability distribution $\pi(\mathbf{x}) \propto \exp(\mathcal{L}(\mathbf{x}))$ and $\mathbf{p}$ be a Gaussian random variable on $\mathcal{P} = \mathbb{R}^d$ with density $p(\mathbf{p}) = \mathcal{N}(\mathbf{p}|\mathbf{0}, \mathbf{M})$ where $\mathbf{M}$ is the covariance matrix. In general, *Hamiltonian Monte Carlo* (HMC) defines a stationary Markov chain on the augmented state space $\mathcal{X} \times \mathcal{P}$ with invariant distribution $p(\mathbf{x}, \mathbf{p}) = \pi(\mathbf{x})p(\mathbf{p})$. The sampler is defined using a *Hamiltonian function*, which up to a constant is the negative log density of $(\mathbf{x}, \mathbf{p})$, given as follows:

$$H(\mathbf{x}, \mathbf{p}) = -\mathcal{L}(\mathbf{x}) + \frac{1}{2}\mathbf{p}^T \mathbf{M}^{-1}\mathbf{p}. \tag{1}$$

In an analogy to physical systems, the first term on the RHS is called the *potential energy*, the second term is called the *kinetic energy*, the state $\mathbf{x}$ is called the *position variable*, and $\mathbf{p}$ the *momentum variable*. Finally, we will call the covariance $\mathbf{M}$ the *mass matrix*. The most common mass matrix is the identity matrix $\mathbf{I}$. Samples in HMC are generated as following. First, the state $\mathbf{p}$ is resampled from its marginal distribution $\mathcal{N}(\mathbf{p}|\mathbf{0}, \mathbf{M})$. Then, given the current state $(\mathbf{x}, \mathbf{p})$, a new state $(\mathbf{x}^*, \mathbf{p}^*)$ is generated by a deterministic simulation of Hamiltonian dynamics:

$$\dot{\mathbf{x}} = \mathbf{M}^{-1}\mathbf{p}; \quad \dot{\mathbf{p}} = -\nabla_{\mathbf{x}}\mathcal{L}(\mathbf{x}). \tag{2}$$

One common approximation to this dynamical system is given by the leapfrog algorithm. One single iteration of leapfrog algorithm is given by the recursive formula

$$\mathbf{p}(\tau + \frac{\epsilon}{2}) = \mathbf{p}(\tau) + \frac{\epsilon}{2}\nabla_{\mathbf{x}}\mathcal{L}(\mathbf{x}(\tau)), \tag{3}$$

$$\mathbf{x}(\tau + \epsilon) = \mathbf{x}(\tau) + \epsilon \mathbf{M}^{-1}\mathbf{p}(\tau + \frac{\epsilon}{2}), \tag{4}$$

$$\mathbf{p}(\tau + \epsilon) = \mathbf{p}(\tau + \frac{\epsilon}{2}) + \frac{\epsilon}{2}\nabla_{\mathbf{x}}\mathcal{L}(\mathbf{x}(\tau + \epsilon)), \tag{5}$$

where $\epsilon$ is the step size and $\tau$ is a discrete time variable. The leapfrog algorithm is initialised by the current sample, that is $(\mathbf{x}(0), \mathbf{p}(0)) = (\mathbf{x}, \mathbf{p})$. After $L$ leapfrog steps (3)-(5), the final state $(\mathbf{x}(L\epsilon), \mathbf{p}(L\epsilon))$ is used as the proposal $(\mathbf{x}^*, \mathbf{p}^*)$ in Metropolis-Hastings correction with acceptance probability $\min[1, \exp(H(\mathbf{x}, \mathbf{p}) - H(\mathbf{x}^*, \mathbf{p}^*))]$. The step size $\epsilon$ and the number of leapfrog steps $L$ are two parameters of HMC.

In many applications, different components of $\mathbf{x}$ may have different scale and be highly correlated. Tuning HMC in such a situation can be very difficult. However, the performance of HMC can be improved by multiplying the state $\mathbf{x}$ by a non-singular matrix $A$. If $A$ is chosen well, the transformed state $\mathbf{x}' = A\mathbf{x}$ may at least locally be better conditioned, i.e., the new variables $\mathbf{x}'$ may be less correlated and have similar scale, so that sampling can be easier. In the context of HMC, this transformation is equivalent to changing mass matrix $\mathbf{M}$. This is because the Hamiltonian dynamics of the system $(A\mathbf{x}, \mathbf{p})$ with mass matrix $\mathbf{M}$ are isomorphic to the dynamics on $(\mathbf{x}, A^T\mathbf{p})$, which is equivalent to defining the state as $(\mathbf{x}, \mathbf{p})$ and using the mass matrix $\mathbf{M}' = A^T\mathbf{M}A$. For a more detailed version of this argument, see the tutorial of Neal [9]. So in this paper we will concentrate on tuning $\mathbf{M}$ on the fly during sampling.

Now, if $\mathcal{L}$ has a constant Hessian $B$ (or nearly so), then a reasonable choice of transformation is to choose $A$ so that $B = AA^T$, because then the Hessian of the log density over $\mathbf{x}'$ will be nearly the identity. This corresponds to a choice of $\mathbf{M} = B$. For more general functions without a constant Hessian, this argument suggests the idea of employing a mass matrix $\mathbf{M}(\mathbf{x})$ that is a function of the position. In this case the Hamiltonian function can be

$$H(\mathbf{x}, \mathbf{p}) = -\mathcal{L}(\mathbf{x}) + \frac{1}{2}\log(2\pi)^d|\mathbf{M}(\mathbf{x})| + \frac{1}{2}\mathbf{p}^T\mathbf{M}(\mathbf{x})^{-1}\mathbf{p}, \tag{6}$$

where the second term on the RHS is from the normalisation factor of Gaussian momentum variable.

## 3   Quasi-Newton Approximations for Sampling

In this section, we describe the Hessian approximation that is used in our samplers. It is based on the well-known BFGS approximation [10], but there are several customizations that we must make to use it within a sampler. First we explain quasi-Newton methods in the context of optimization. Consider minimising the function $f : \mathbb{R}^d \to \mathbb{R}$, quasi-Newton methods search for the minimum of $f(\mathbf{x})$ by generating a sequence of iterates $\mathbf{x}_{k+1} = \mathbf{x}_k - \alpha_k \mathbf{H}_k \nabla f(\mathbf{x}_k)$ where $\mathbf{H}_k$ is an approximation to the inverse Hessian at $\mathbf{x}_k$, which is computed from the previous function values and gradients. One of the most popular large scale quasi-Newton methods is *limited-Memory BFGS (L-BFGS)* [10]. Given the previous $m$ iterates $\mathbf{x}_{k-m+1}, \mathbf{x}_{k-m+2}, \ldots \mathbf{x}_k$, the L-BFGS approximation $\mathbf{H}_{k+1}$ is

$$\mathbf{H}_{k+1} = (\mathbf{I} - \frac{\mathbf{y}_k \mathbf{s}_k^T}{\mathbf{s}_k^T \mathbf{y}_k}) \mathbf{H}_k (\mathbf{I} - \frac{\mathbf{s}_k \mathbf{y}_k^T}{\mathbf{s}_k^T \mathbf{y}_k}) + \mathbf{s}_k \mathbf{s}_k^T \tag{7}$$

where $\mathbf{s}_k = \mathbf{x}_{k+1} - \mathbf{x}_k$ and $\mathbf{y}_k = \nabla f_{k+1} - \nabla f_k$. The base case of the recursion is typically chosen as $\mathbf{H}_{k-m} = \gamma \mathbf{I}$ for some $\gamma \in \mathbb{R}$. If $m = k$, then this is called the *BFGS* formula, and typically it is implemented by storing the full $d \times d$ matrix $\mathbf{H}_k$. If $m < k$, however, this is called *limited-memory BFGS*, and can be implemented much more efficiently. It can be seen that the BFGS formula (7) is a rank-two update to the previous Hessian approximation $\mathbf{H}_k$. Therefore $\mathbf{H}_{k+1}$ is a diagonal matrix plus a rank $2m$ matrix, so the matrix vector product $\mathbf{H}_k \nabla f(\mathbf{x}_k)$ can be computed in linear time $O(md)$. Typically the product $\mathbf{Hv}$ is implemented by a special two-loop recursive algorithm [10].

In contrast to optimization methods, most sampling methods need a factorized form of $\mathbf{H}_k$ to draw samples from $\mathcal{N}(0, \mathbf{H}_k)$. More precisely, we adopt the factorisation $\mathbf{H}_k = \mathbf{S}_k \mathbf{S}_k^T$, so that we can generate a sample as $\mathbf{p} = \mathbf{S}_k \mathbf{z}$ where $\mathbf{z} \sim \mathcal{N}(\mathbf{0}, \mathbf{I})$. The matrix operations to obtain $\mathbf{S}_k$, e.g. the Cholesky decomposition cost $\mathcal{O}(d^3)$. To avoid this cost, we need a way to compute $\mathbf{S}_k$ that does not require constructing the matrix $\mathbf{H}_k$ explicitly. Fortunately there is a variant of the BFGS formula that maintains $\mathbf{S}_k$ directly [2], which is

$$\mathbf{H}_{k+1} = \mathbf{S}_{k+1} \mathbf{S}_{k+1}^T; \quad \mathbf{S}_{k+1} = \left(\mathbf{I} - \mathbf{p}_k \mathbf{q}_k^T\right) \mathbf{S}_k \tag{8}$$

$$\mathbf{B}_{k+1} = \mathbf{C}_{k+1} \mathbf{C}_{k+1}^T; \quad \mathbf{C}_{k+1} = \left(\mathbf{I} - \mathbf{u}_k \mathbf{t}_k^T\right) \mathbf{C}_k \tag{9}$$

$$\mathbf{p}_k = \frac{\mathbf{s}_k}{\mathbf{s}_k^T \mathbf{y}_k}; \quad \mathbf{q}_k = \sqrt{\frac{\mathbf{s}_k^T \mathbf{y}_k}{\mathbf{s}_k^T \mathbf{B}_k \mathbf{y}_k}} \mathbf{B}_k \mathbf{s}_k - \mathbf{y}_k \tag{10}$$

$$\mathbf{t}_k = \frac{\mathbf{s}_k}{\mathbf{s}_k^T \mathbf{B}_k \mathbf{s}_k}; \quad \mathbf{u}_k = \sqrt{\frac{\mathbf{s}_k^T \mathbf{B}_k \mathbf{s}_k}{\mathbf{s}_k^T \mathbf{y}_k}} \mathbf{y}_k + \mathbf{B}_k \mathbf{s}_k \tag{11}$$

where $\mathbf{B}_k = \mathbf{H}_k^{-1}$ denotes the Hessian matrix approximation. Again, we will use a limited-memory version of these updates, in which the recursion is stopped at $\mathbf{H}_{k-m} = \gamma \mathbf{I}$.

As for the running time of the above approximation, computing $\mathbf{S}_k$ requires $O(m^2 d)$ time and $O(md)$ space, so it is still linear in the dimensionality. The matrix vector product $\mathbf{S}_{k+1} \mathbf{z}$ can be computed by a sequence of inner products $\mathbf{S}_{k+1} \mathbf{z} = \prod_{i=k-m-1}^{k} (\mathbf{I} - \mathbf{p}_i \mathbf{q}_i^T) \mathbf{S}_{k-m} \mathbf{z}$, in time $O(md)$.

A second issue is that we need $\mathbf{H}_k$ to be positive definite if it is to be used as a covariance matrix. It can be shown [10] that $\mathbf{H}_k$ is positive definite if for all $i \in (k-m+1, k)$, we have $\mathbf{s}_i^T \mathbf{y}_i > 0$. For a convex function $f$, an optimizer can be arranged so that this condition always holds, but we cannot do this in a sampler. Instead, we first sort the previous samples $\{\mathbf{x}_i\}$ in ascending order with respect to $\mathcal{L}(\mathbf{x})$, and then check if there are any adjacent pairs $(\mathbf{x}_i, \mathbf{x}_{i+1})$ such that the resulting $\mathbf{s}_i$ and $\mathbf{y}_i$ have $\mathbf{s}_i^T \mathbf{y}_i \leq 0$. If this happens, we remove the point $\mathbf{x}_{i+1}$ from the memory and recompute $\mathbf{s}_i, \mathbf{y}_i$ using $\mathbf{x}_{i+2}$, and so on. In this way we can ensure that $\mathbf{H}_k$ is always positive definite.

Although we have described BFGS as relying on a memory of "previous" points, e.g., previous iterates of an optimization algorithm, or previous samples of an MCMC chain, in principle the BFGS equations could be used to generate a Hessian approximation from any set of points $\mathbf{X} = \{\mathbf{x}_1 \ldots \mathbf{x}_m\}$. To emphasize this, we will write $\mathbf{H}_{\text{BFGS}} : \mathbf{X} \mapsto \mathbf{H}_k$ for the function that maps a "pseudo-memory" $\mathbf{X}$ to the inverse Hessian $\mathbf{H}_k$. This function first sorts $\mathbf{x} \in \mathbf{X}$ by $\mathcal{L}(\mathbf{x}_i)$, then computes $\mathbf{s}_i = \mathbf{x}_{i+1} - \mathbf{x}_i$ and $\mathbf{y}_i = \nabla \mathcal{L}(\mathbf{x}_{i+1}) - \nabla \mathcal{L}(\mathbf{x}_i)$, then filters $\mathbf{x}_i$ as described above so that $\mathbf{s}_i^T \mathbf{y}_i > 0 \, \forall i$, and finally computes the Hessian approximation $\mathbf{H}_k$ using the recursion (8)–(11).

# 4 Quasi-Newton Markov Chain Monte Carlo

In this section, we describe two new quasi-Newton samplers. They will both follow the same structure, which we describe now. Intuitively, we want to use the characteristics of the target distribution to accelerate the exploration of the region with high probability mass. The previous samples provide information about the target distribution, so it is reasonable to use them to adapt the kernel. However, naively tuning the sampling parameters using all previous samples may lead to an invalid chain, that is, a chain that does not have $\pi$ as its invariant distribution.

Our samplers will use a simple solution to this problem. Rather than adapting the kernel using all of the previous samples in the Markov chain, we will adapt using a limited window of $K$ previous samples. The chain as a whole will then be an order $K$ Markov chain. It is easiest to analyze this chain by converting it into a first-order Markov chain over an enlarged space. Specifically, we build a Markov chain in a $K$-fold product space $\mathcal{X}^K$ with the stationary distribution $p(\mathbf{x}_{1:K}) = \prod_{i=1:K} \pi(\mathbf{x}_i)$. We denote a state of this chain by $\mathbf{x}_{t-K+1}, \mathbf{x}_{t-K+2}, \ldots, \mathbf{x}_t$. We use the short-hand notation $\mathbf{x}_{1:K\setminus i}^{(t)}$ for the subset of $\mathbf{x}_{1:K}^{(t)}$ excluding the $\mathbf{x}_i^{(t)}$.

Our samplers will then update one component of $\mathbf{x}_{1:K}^{(t)}$ per iteration, in a Gibbs-like fashion. We define a transition kernel $T_i$ that only updates the $i$th component of $\mathbf{x}_{1:K}^{(t)}$, that is:

$$T_i(\mathbf{x}_{1:K}^{(t)}, \mathbf{x}_{1:K}') = \delta(\mathbf{x}_{1:K\setminus i}^{(t)}, \mathbf{x}_{1:K\setminus i}')B(\mathbf{x}_i, \mathbf{x}_i'|\mathbf{x}_{1:K\setminus i}^{(t)}), \tag{12}$$

where $B(\mathbf{x}_i, \mathbf{x}_i'|\mathbf{x}_{1:K\setminus i})$ is called the *base kernel* that is a MCMC kernel in $\mathcal{X}$ and adapts with $\mathbf{x}_{1:K\setminus i}^{(t)}$. If $B$ leaves $\pi(\mathbf{x}_i)$ invariant for all fixed values of $\mathbf{x}_{1:K\setminus i}$, it is straightforward to show that $T_i$ leaves $p$ invariant. Then, the sampler as a whole updates each of the components $\mathbf{x}_i^{(t)}$ in sequence, so that the method as a whole is described by the kernel

$$T(\mathbf{x}_{1:K}, \mathbf{x}_{1:K}') = T_1 \circ T_2 \ldots \circ T_K(\mathbf{x}_{1:K}, \mathbf{x}_{1:K}'), \tag{13}$$

where $T_i \circ T_j$ denotes composition of kernels $T_i$ and $T_j$. Because the each kernel $T_i$ leaves $p(\mathbf{x}_{1:K})$ invariant, the composition kernel $T$ also leaves $p(\mathbf{x}_{1:K})$ invariant. Such an adaptive scheme is equivalent to using an ensemble of $K$ chains and changing the kernel of each chain with the state of the others. It is called the *ensemble-chain adaptation* (ECA) in this paper. One early example of ECA is found in [4]. To simplify the analysis of the validity of the chain, we assume the base kernel $B$ is irreducible in one iteration. This assumption can be satisfied by many popular MCMC kernels.

## 4.1 Using BFGS within Metropolis-Hastings

A simple way to incorporate quasi-Newton approximations within MCMC is to use the Metropolis-Hastings (M-H) algorithm. The intuition is to fit the Gaussian proposal distribution to the target distribution, so that points in a high probability region are more likely to be proposed. We will call this algorithm MHBFGS. Specifically, the proposal distribution of MHBFGS is defined as $q(\mathbf{x}'|\mathbf{x}_{1:K}^{(t)}) = \mathcal{N}(\mathbf{x}'; \mu, \Sigma)$, where the proposal mean $\mu = \mu(\mathbf{x}_{1:K}^{(t)})$ and covariance $\Sigma = \Sigma(\mathbf{x}_{1:K}^{(t)})$ depend on the state of all $K$ chains.

Several choices for the mean function are possible. One simple choice is to use one of the samples in the window as the mean, e.g., $\mu(\mathbf{x}_{1:K}^{(t)}) = \mathbf{x}_1^{(t)}$. Another potentially better choice is a Newton step from $\mathbf{x}_t$. For the covariance function at $\mu$, we will use the BFGS approximation $\Sigma(\mathbf{x}_{1:K}) = \mathbf{H}_{\text{BFGS}}(\mathbf{x}_{1:K})$. The proposal $\mathbf{x}'$ of $T_1$ is accepted with probability

$$\alpha(\mathbf{x}_1^{(t)}, \mathbf{x}') = \min\left(1, \frac{q(\mathbf{x}_1^{(t)}|\mathbf{x}_{2:K}^{(t)}, \mathbf{x}')}{\pi(\mathbf{x}_1^{(t)})} \frac{\pi(\mathbf{x}')}{q(\mathbf{x}'|\mathbf{x}_1^{(t)}, \mathbf{x}_{2:K}^{(t)})}\right). \tag{14}$$

If $\mathbf{x}'$ is rejected, $\mathbf{x}_1^{(t)}$ is duplicated as the new sample. Because the Gaussian proposal $q(A|\mathbf{x}_1^{(t)}, \mathbf{x}_{2:K}^{(t)})$ has positive probability for all $A \in \mathcal{X}$, the M-H kernel is irreducible within one iteration. Because the M-H algorithm with acceptance ratio defined as (14) leaves $\pi(\mathbf{x})$ invariant, MHBFGS is a valid method that leaves $p(\mathbf{x}_{1:K})$ invariant. Although MHBFGS is simple and intuitive, in preliminary experiments we have found that MHBFGS sampler may converge slowly in high

**Algorithm 1**

HMCBFGS

**Input:** Current memory $(\mathbf{x}_1^{(t)}, \mathbf{x}_2^{(t)}, \ldots, \mathbf{x}_K^{(t)})$

**Output:** Next memory $(\mathbf{x}_1^{(t+1)}, \mathbf{x}_2^{(t+1)}, \ldots, \mathbf{x}_K^{(t+1)})$

1: $\mathbf{p} \sim \mathcal{N}(0, \mathbf{B}_{\text{BFGS}}(\mathbf{x}_{2:K}^{(t)}))$
2: $(\mathbf{x}^*, \mathbf{p}^*) \leftarrow \text{Leapfrog}(\mathbf{x}_1^{(t)}, \mathbf{p})$ using (3)-(5)
3: $u \sim \text{Unif}[0, 1]$
4: **if** $u \leq \exp(H(\mathbf{x}_1^{(t)}, \mathbf{p}|\mathbf{x}_{2:K}) - H(\mathbf{x}^*, \mathbf{p}^*|\mathbf{x}_{2:K}))$ **then**
5:     $\mathbf{x}_K^{(t+1)} \leftarrow \mathbf{x}^*$
6: **else**
7:     $\mathbf{x}_K^{(t+1)} \leftarrow \mathbf{x}_K^{(t)}$
8: **end if**
9: $\mathbf{x}_{1:K-1}^{(t+1)} \leftarrow \mathbf{x}_{2:K}^{(t)}$
10: **return** $(\mathbf{x}_1^{(t+1)}, \mathbf{x}_2^{(t+1)}, \ldots, \mathbf{x}_K^{(t+1)})$

dimensions. In general Metropolis Hastings with a Gaussian proposal can suffer from random walk behavior, even if the true Hessian is used. For this reason, next we incorporate the BFGS into a more sophisticated sampling algorithm.

## 4.2 Using BFGS within Hamiltonian Monte Carlo

Better convergence speed can be achieved by incorporating BFGS within the HMC kernel. The high-level idea is to start with the MHBFGS algorithm, but to replace the Gaussian proposal with a simulation of Hamiltonian dynamics. However, we will need to be a bit careful in order to ensure that the Hamiltonian is separable, because otherwise we would need to employ a generalized leapfrog integrator [5] which is significantly more expensive.

The new samples in HMCBFGS are generated as follows. As before we update one component of $\mathbf{x}_{1:K}^{(t)}$ at a time. Say that we are updating component $i$. First we sample a new value of the momentum variable $\mathbf{p} \sim \mathcal{N}(0, \mathbf{B}_{\text{BFGS}}(\mathbf{x}_{1:K\setminus i}^{(t)}))$. It is important that when constructing the BFGS approximation, we *not* use the value $\mathbf{x}_i^{(t)}$ that we are currently resampling. Then we simulate the Hamiltonian dynamics starting at the point $(\mathbf{x}_i^{(t)}, \mathbf{p})$ using the leapfrog method (3)–(5). The Hamiltonian energy used for this dynamics is simply

$$H_i(\mathbf{x}_{1:K}^{(t)}, \mathbf{p}) = -\mathcal{L}(\mathbf{x}_i^{(t)}) + \frac{1}{2}\mathbf{p}^T\mathbf{H}_{\text{BFGS}}(\mathbf{x}_{1:K\setminus i}^{(t)})^{-1}\mathbf{p}, \qquad (15)$$

This yields a proposed value $(\mathbf{x}^*, \mathbf{p}^*)$. Finally, the proposal is accepted with probability $\min[1, \exp(H(\mathbf{x}_i, \mathbf{p}) - H(\mathbf{x}_i^*, \mathbf{p}^*)]$, for $H$ in (15) and $\mathbf{p}^*$ is discarded after M-H correction. This procedure is summarized in Algorithm 1.

This procedure is an instance of the general ECA scheme described above, with base kernel

$$B(\mathbf{x}_i, \mathbf{x}_i'|\mathbf{x}_{1:K\setminus i}) = \int \hat{B}(\mathbf{x}_i, \mathbf{p}_i, \mathbf{x}_i', \mathbf{p}_i'|\mathbf{x}_{1:K\setminus i})d\mathbf{p}_i d\mathbf{p}_i'.$$

where $\hat{B}(\mathbf{x}_i, \mathbf{p}_i, \mathbf{x}_i', \mathbf{p}_i'|\mathbf{x}_{1:K\setminus i})$ is a standard HMC kernel with mass matrix $\mathbf{B}_{\text{BFGS}}(\mathbf{x}_{1:K\setminus i})$ that includes sampling $\mathbf{p}_i$. The Hamiltonian energy function of $\hat{B}$ given by (15) is *separable*, that means $\mathbf{x}_i$ only appear in potential energy. It is easy to see that $B$ is a valid kernel in $\mathcal{X}$, so as a ECA method, HMCBFGS leaves $p(\mathbf{x}_{1:K}) = \prod_i \pi(x_i)$ invariant.

It is interesting to consider if the method is valid in the augmented space $\mathcal{X}^K \times \mathcal{P}^K$, i.e., whether Algorithm 1 leaves the distribution

$$p(\mathbf{x}_{1:K}, \mathbf{p}_{1:K}) = \prod_{i=1}^K \pi(\mathbf{x}_i)\mathcal{N}(\mathbf{p}_i; 0, \mathbf{B}_{\text{BFGS}}(\mathbf{x}_{1:K\setminus i}^{(t)}))$$

Interestingly, this is *not* true, because every update to $\mathbf{x}_i$ changes the Gaussian factors for the momentum variables $\mathbf{p}_j$ for $j \neq i$ in a way that the Metropolis Hastings correction in lines 4–8 does not consider. So despite the auxiliary variables, it is easiest to establish validity in the original space.

HMCBFGS has the advantages of being a simple approach that only uses gradient and the computational efficiency that the cost of all matrix operations (namely in lines 1 and 2 of Algorithm 1) is at the scale of $O(Kd)$. But, being an ECA method, HMCBFGS has the disadvantage that the larger the number of chains $K$, the updates are "spread across" the chains, so that each chain gets a small number of updates during a fixed amount of computation time. In Section 6 we will evaluate empirically whether this potential drawback is outweighed by the advantages of using approximate second-order information.

## 5   Related Work

Girolami and Calderhead [5] propose a new HMC method called *Riemannian manifold Hamiltonian Monte Carlo* (RMHMC) where $\mathbf{M}(\mathbf{x})$ can be any positive definite matrix. In their work, $\mathbf{M}(\mathbf{x})$ is chosen to be the expected Fisher information matrix and the experimental results show that RMHMC can converge much faster than many other MCMC methods. Girolami and Calderhead adopted a generalised leapfrog method that is a reversible and volume-preserving approximation to non-separable Hamiltonian. However, such a method may require computing *third-order* derivatives of $\mathcal{L}$, which can be infeasible in many applications.

Barthelme and Chopin [1] pointed out the possibility to use approximate BFGS Hessian in RMHMC for computational efficiency. Similarly, Roy [14] suggested iteratively updating the local metric approximation. Roy also emphasized the potential effect of such an iterative approximation to the validity, a main problem that we address here. An early example of ECA is *adaptive direction sampling* (ADS) [4], in which each sample is taken along a random direction that is chosen based on the samples from a set of chains. However, the validity of ADS can be established only when the size of ensemble is greater than the number of dimensions, otherwise the samples are trapped in a subspace. HMCBFGS avoids this problem because the BFGS Hessian approximation is full rank.

There has been a large amount of interest in *adaptive* MCMC methods that accumulate information from all previous samples. These methods must be designed carefully because if the kernel is adapted with the full sampling history in a naive way, the sampler can be invalid [13]. A well known example of a correct adaptive algorithm is the Adaptive Metropolis [6] algorithm, which adapts the Gaussian proposal of a Metropolis Hastings algorithm based on the empirical covariance of previous samples in a way that maintains ergodicity. Being a valid method, the adaptation of kernel must keep decreasing over time. In practice, the parameters of the kernel in many diminishing adaptive methods converge to a single value over the entire state space. This could be problematic if we want the sampler to adapt to local characteristics of the target distribution, e.g., if different regions of the target distribution have different curvature. Using a finite memory of recent samples, our method avoids such a problem.

## 6   Experiments

We test HMCBFGS on two different models, Bayesian logistic regression and Bayesian conditional random fields (BCRFs). We compare HMCBFGS to the standard HMC which uses identity mass matrix and RMHMC which requires computing the Hessian matrix. All methods are implemented in Java [1]. We do not report results from MHBFGS because preliminary experiments showed that it was much worse than either HMC or HMCBFGS. The datasets for Bayesian logistic regression is used for RMHMC in [5]. For HMC and HMCBFGS we employ the random step size $\epsilon \sim \mathrm{Unif}[0.9\hat{\epsilon}, \hat{\epsilon}]$, where $\hat{\epsilon}$ is the maximum step size. For RMHMC, we used the fixed $\epsilon = 0.5$ for all datasets that follows the setting in [5].

For HMC and HMCBFGS we tuned $L$ on one data set (the German data set) and used that value on all datasets. We chose the smallest number of leaps that did not degrade the performance of the sampler. $L$ was chosen to be $40$ for HMC and to be $20$ for HMCBFGS. For RMHMC, we employed $L = 6$ leaps in RMHMC, following Girolami and Calderhead [5]. For HMCBFGS, we heuristically chose the number of ensemble chains $K$ to be slightly higher than $d/2$.

For each method, we drew 5000 samples after 1000 burn-in samples. The convergence speed is measured by effective sample size (ESS) [5], which summaries the amount of autocorrelation across different lags over all dimensions[2]. The more detailed description of ESS can be found in [5]. Because HMCBFGS displays more correlation within individual chain than across chains, we calculate the ESS separately for individual chains in the ensemble and the overall ESS is simply the sum of that from individual chains. All the final ESS on each data set is obtained by averaging over 10 runs using different initialisations.

| ESS | HMC | HMCBFGS | RMHMC |
|---|---|---|---|
| Min | 3312 | 3643 | **4819** |
| Mean | 3862 | 4541 | **4950** |
| Max | 4445 | 4993 | **5000** |
| Time (s) | 7.56 | **4.74** | 483.00 |
| ES/s | 739 | **1470** | 107 |

Table 1: Performance of MCMC samplers on Bayesian logistic regression, as measured by Effective Sample Size (ESS). Higher is better. Averaged over five datasets. ES/s is the number of effective samples per second

| Dataset | D | N | HMC | HMCBFGS | RMHMC |
|---|---|---|---|---|---|
| Australian | 15 | 690 | 396 | **818** | 18 |
| German | 25 | 1000 | 255 | **397** | 3 |
| Heart | 14 | 532 | 1054 | **2009** | 54 |
| Pima | 8 | 270 | 591 | **1383** | 118 |
| Ripley | 7 | 250 | 1396 | **2745** | 344 |

Table 2: Effective samples per second on Bayesian logistic regression. D is the number of regression coefficients and N is the size of training data set

The results on ESS averaged over five datasets on Bayesian logistic regression are given by Table 1. Our ESS number of HMC and RMHMC basically replicates the results in [5]. RMHMC achieves the highest minimum and mean and maximum ESS and that are all very close to the total number of samples 5000. However, because HMC and our method only require computing the gradient, they outperforms RMHMC in terms of mean ESS per second. HMCBFGS gains a 10%, 17% and 12% increase in minimum, mean and maximum ESS than HMC, but only needs half number of leaps for HMC. A detailed performance of methods over datasets is shown in Table 2.

The second model that we use is a Bayesian CRF on a small natural language dataset of FAQs from Usenet [8]. A linear-chain CRF is used with Gaussian prior on the parameters. The model has 120 parameters. This model has been used previously [12, 15]. In a CRF it is intractable to compute the Hessian matrix exactly, so RMHMC is infeasible. For HMCBFGS we use $K = 5$ ensemble chains. Each method is also tested 10 times with different initial points. For each chain we draw 8000 samples with 1000 burn-in. We use the step size $\epsilon = 0.02$ and the number of leaps $L = 10$ for both HMC and HMCBFGS. This parameter setting gives 84% acceptance rate on both HMC and HMCBFGS (averaged over the 10 runs).

Figure 1 shows the sample trajectory plots for HMC and HMCBFGS on seven randomly selected dimensions. It is clear that HMCBFGS demonstrates remarkably less autocorrelation than HMC. The statistics of ESS in Table 3 gives a quantitative evaluation of the performance of HMC and HMCBFGS. The results suggest that BFGS approximation dramatically reduces the sample autocorrelation with a small increase of computational overhead on this dataset.

Finally, we evaluate the scalability of the methods on the highly correlated 1000 dimensional Gaussian $\mathcal{N}(\mathbf{0}, \mathbf{1}\mathbf{1}^T + 4)$. Using an ensemble of $K = 5$ chains, the samples from HMCBFGS are less correlated than HMC along the largest eigenvalue direction (Figure 2).

| ESS | HMC | HMCBFGS |
|---|---|---|
| Min | 3 | **26** |
| Mean | 9 | **438** |
| Max | 25 | **5371** |
| Time (s) | **35743** | 37387 |
| ES/h | 1 | **42** |

Table 3: Performance of MCMC samplers on Bayesian CRFs, as measured by Effective Sample Size (ESS). Higher is better. ES/h is the number of effective samples per hour

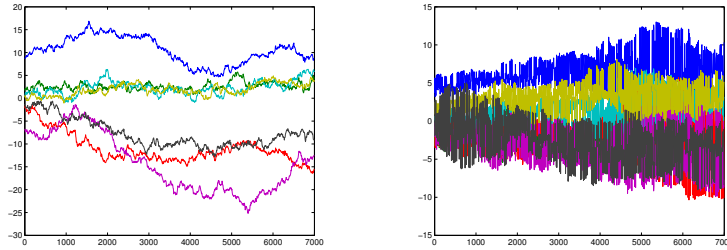

Figure 1: Sample trace plot of 7000 samples from the posterior of a Bayesian CRF using HMC (left) and our method HMCBFGS (rigt) from a single run of each sampler (each line represents a dimension)

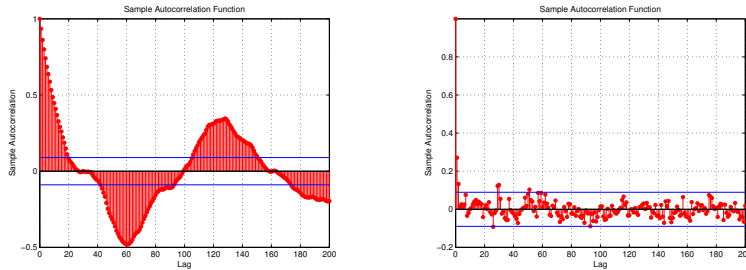

Figure 2: ACF plot of samples projected on to the direction of largest eigenvector of 1000 dimensional Gaussian using HMC(left) and HMCBFGS(right)

## 7  Discussion

To the best of our knowledge, this paper presents the first adaptive MCMC methods to employ quasi-Newton approximations. Naive attempts at combining these ideas (such as MHBFGS) do not work well. On the other hand, HMCBFGS is more effective than the state-of-the-art sampler on several real world data sets. Furthermore, HMCBFGS works well on a high dimensional model, where full second-order methods are infeasible, with little extra overhead over regular HMC.

As far as future work, our current method may not work well in regions where the density is not convex, because the true Hessian is not positive definite. Another potential issue, the asymptotic independence between the chains in ECA methods may lead to poor Hessian approximations. On a brighter note, our work raises the interesting possibility that quasi-Newton methods, which are almost exclusively used within the optimization literature, may be useful more generally.

#### Acknowledgments

We thank Iain Murray for many useful discussions, and Mark Girolami for detailed comments on an earlier draft.

## Footnotes

[1] Our implementation was based on the Matlab code of RMHMC of Girolami and Calderhead and checked against the original Matlab version

[2]We use the code from [5] to compute ESS of samples

# References

[1] S. Barthelme and N. Chopin. Discussion on Riemannian Manifold Hamiltonian Monte Carlo. *Journal of the Royal Statistical Society, B (Statistical Methodology)*, 73:163–164, 2011. doi: 10.1111/j.1467-9868.2010.00765.x.

[2] K. Brodlie, A. Gourlay, and J. Greenstadt. Rank-one and rank-two corrections to positive definite matrices expressed in product form. *IMA Journal of Applied Mathematics*, 11(1): 73–82, 1973.

[3] S. Chib, E. Greenberg, and R. Winkelmann. Posterior simulation and bayes factors in panel count data models. *Journal of Econometrics*, 86(1):33–54, June 1998. URL `http://ideas.repec.org/a/eee/econom/v86y1998i1p33-54.html`.

[4] W. R. Gilks, G. O. Roberts, and E. I. George. Adaptive direction sampling. *The Statistician*, 43(1):179–9, 1994.

[5] M. Girolami and B. Calderhead. Riemannian manifold Hamiltonian Monte Carlo (with discussion). *Journal of the Royal Statistical Society, B (Statistical Methodology)*, 73:123–214, 2011. doi: 10.1111/j.1467-9868.2010.00765.x.

[6] H. Haario, E. Saksman, and J. Tamminen. An adaptive Metropolis algorithm. *Bernoulli*, 7(2): 223–242, 2001.

[7] J. S. Liu, F. Liang, and W. H. Wong. The multiple-try method and local optimization in Metropolis sampling. *Journal of the American Statistical Association*, 95(449):pp. 121–134, 2000.

[8] A. McCallum. Frequently asked questions data set. `http://www.cs.umass.edu/~mccallum/data/faqdata`.

[9] R. M. Neal. MCMC using Hamiltonian dynamics. In S. Brooks, A. Gelman, G. Jones, and X.-L. Meng, editors, *Handbook of Markov Chain Monte Carlo*. Chapman & Hall / CRC Press, 2010.

[10] J. Nocedal and S. J. Wright. *Numerical Optimization*. Springer-Verlag, New York, 1999. ISBN 0-387-98793-2.

[11] Y. Qi and T. P. Minka. Hessian-based Markov chain Monte Carlo algorithms. In *First Cape Cod Workshop on Monte Carlo Methods*, September 2002.

[12] Y. Qi, M. Szummer, and T. P. Minka. Bayesian conditional random fields. In *Artificial Intelligence and Statistics (AISTATS)*. Barbados, January 2005.

[13] G. O. Roberts and J. S. Rosenthal. Coupling and ergodicity of adaptive mcmc. *Journal of Applied Probability*, 44(2):458–475, 2007.

[14] D. M. Roy. Discussion on Riemannian Manifold Hamiltonian Monte Carlo. *Journal of the Royal Statistical Society, B (Statistical Methodology)*, 73:194–195, 2011. doi: 10.1111/j.1467-9868.2010.00765.x.

[15] M. Welling and S. Parise. Bayesian random fields: The Bethe-Laplace approximation. In *Uncertainty in Artificial Intelligence (UAI)*, 2006.

